# A Stochastic approximation method for inference in probabilistic graphical models

**Peter Carbonetto**
Dept. of Human Genetics
University of Chicago
Chicago, IL, U.S.A.
pcarbone@bsd.uchicago.edu

**Matthew King**
Dept. of Botany
University of British Columbia
Vancouver, B.C., Canada
kingdom@interchange.ubc.ca

**Firas Hamze**
D-Wave Systems
Burnaby, B.C., Canada
fhamze@dwavesys.com

## Abstract

We describe a new algorithmic framework for inference in probabilistic models, and apply it to inference for latent Dirichlet allocation (LDA). Our framework adopts the methodology of variational inference, but unlike existing variational methods such as mean field and expectation propagation it is not restricted to tractable classes of approximating distributions. Our approach can also be viewed as a "population-based" sequential Monte Carlo (SMC) method, but unlike existing SMC methods there is no need to design the artificial sequence of distributions. Significantly, our framework offers a principled means to exchange the variance of an importance sampling estimate for the bias incurred through variational approximation. We conduct experiments on a difficult inference problem in population genetics, a problem that is related to inference for LDA. The results of these experiments suggest that our method can offer improvements in stability and accuracy over existing methods, and at a comparable cost.

## 1 Introduction

Over the past several decades, researchers in many different fields—statistics, economics, physics, genetics and machine learning—have focused on coming up with more accurate and more efficient approximate solutions to intractable probabilistic inference problems. To date, there are three widely-explored approaches to approximate inference in probabilistic models: obtaining a Monte Carlo estimate by simulating a Markov chain (MCMC); obtaining a Monte Carlo estimate by drawing samples from a distribution other than the target then reweighting the samples to account for any discrepancies (importance sampling); and variational inference, in which the original integration problem is transformed into an optimization problem.

The variational approach in particular has attracted wide interest in the machine learning community, and this interest has lead to a number of important innovations in approximate inference—some of these more recent developments are described in the dissertations of Beal [3], Minka [22], Ravikumar [27] and Wainwright [31]. The key idea behind variational inference is to come up with a family of approximating distributions $\hat{p}(x; \theta)$ that have "nice" analytic properties, then to optimize some criterion in order to find the distribution parameterized by $\theta$ that most closely matches the target posterior $p(x)$. All variational inference algorithms, including belief propagation and its generalizations [32], expectation propagation [22] and mean field [19], can be derived from a common objective, the Kullback-Leibler (K-L) divergence [9]. The major drawback of variational methods is that the best approximating distribution may still impose an unrealistic or questionable factorization, leading to excessively biased estimates (see Fig. 1, left-hand side).

In this paper, we describe a new variational method that does not have this limitation: it adopts the methodology of variational inference without being restricted to tractable classes of approximate

distributions (see Fig. 1, right-hand side). The catch is that the variational objective (the K-L divergence) is difficult to optimize because its gradient cannot be computed exactly. So to descend along the surface of the variational objective, we propose to employ stochastic approximation [28] with Monte Carlo estimates of the gradient, and update these estimates over time with sequential Monte Carlo (SMC) [12]—hence, *a stochastic approximation method for probabilistic inference*. Large gradient descent steps may quickly lead to a degenerate sample, so we introduce a mechanism that safeguards the variance of the Monte Carlo estimate at each iteration (Sec. 3.5). This variance safeguard mechanism does not make the standard *effective sample size* (ESS) approximation [14], hence it is likely to more accurately monitor the variance of the sample.

Indirectly, the variance safeguard provides a way to obtain an estimator that has low variance in exchange for (hopefully small) bias. To our knowledge, our algorithm is the first general means of achieving such a trade-off and, in so doing, it draws meaningful connections between Monte Carlo and variational methods.

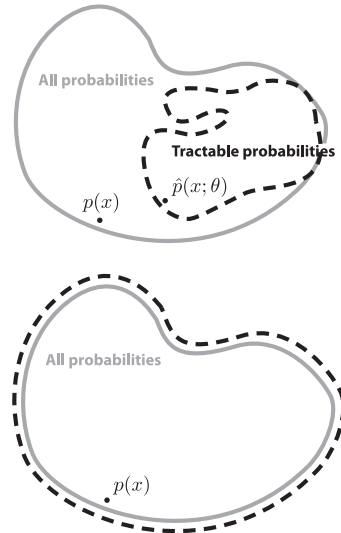

The advantage of our stochastic approximation method with respect to other variational methods is rather straightforward: it does not restrict the class of variational densities by making assumptions about their structure. However, whe advantage of our approach compared to Monte Carlo methods such as annealed importance sampling (AIS) [24] is less obvious. One key advantage is that there is no need to design the sequence of SMC distributions as it is a direct product of the algorithm's derivation (Sec. 3). It is our conjecture that this automatic selection, when combined with the variance safeguard, is more efficient than setting the sequence by hand, say, via tempered transitions [12, 18, 24]. The population genetics experiments we conduct in Sec. 4 provide some support for this claim.

We illustrate our approach on the problem of inferring population structure from a cohort of genotyped sequences using the mixture model of Pritchard et al. [26]. We show in Sec. 4 that Markov chain Monte Carlo (MCMC) is prone to producing very different answers in independent simulations, and that it fails to adequately capture the uncertainty in its solutions. For many population genetics applications, such as wildlife conservation [8], it is crucial to accurately characterize the confidence in a solution. Since variational methods employing mean field approximations [4, 30] tend to be overconfident, they are poorly suited for this problem. (This has generally not been an issue for semantic text analysis [4, 15].) As we show, SMC with a

Figure 1: The guiding principle behind standard variational methods *(top)* is to find the approximating density $\hat{p}(x; \theta)$ that is closest to the distribution of interest $p(x)$, yet remains within the defined set of tractable probability distributions. In our approach *(bottom)*, the class of approximating densities always coincides with the target $p(x)$.

uniform sequence of tempered distributions fares little better than MCMC. The implementation of our approach on the population structure model demonstrates improvements in both accuracy and reliability over MCMC and SMC alternatives, and at a comparable computational cost.

The latent Dirichlet allocation (LDA) model [4] is very similar to the population structure model of [26], under the assumption of fixed Dirichlet priors. Since LDA is already familiar to the machine learning audience, it serves as a running example throughout our presentation.

## 1.1 Related work

The interface of optimization and simulation strategies for inference has been explored in a number of papers, but none of the existing literature resembles the approach proposed in this paper. De Freitas et al. [11] use a variational approximation to formulate a Metropolis-Hastings proposal. Recent work on adaptive MCMC [1] combines ideas from both stochastic approximation and MCMC to automatically learn better proposal distributions. Our work is also unrelated to the paper [20] with a similar title, where stochastic approximation is applied to improving the Wang-Landau algorithm. Younes [33] employs stochastic approximation to compute the maximum likelihood estimate of an undirected graphical model. Also, the cross-entropy method [10] uses importance sampling and optimization for inference, but exhibits no similarity to our work beyond that.

## 2 Latent Dirichlet allocation

Latent Dirichlet allocation (LDA) is a generative model of a collection of text documents, or *corpus*. Its two key features are: the order of the words is unimportant, and each document is drawn from a mixture of topics. Each document $d = 1, \dots, D$ is expressed as a "bag" of words, and each word $w_{di} = j$ refers to a vocabulary item $j \in \{1, \dots, W\}$. (Here we assume each document has the same length $N$.) Also, each word has a latent topic indicator $z_{di} \in \{1, \dots, K\}$. Observing the $j$th vocabulary item in the $k$th topic occurs with probability $\beta_{kj}$. The word proportions for each topic are generated according to a Dirichlet distribution with fixed prior $\eta$. The latent topic indicators are generated independently according to $p(z_{di} = k \,|\, \tau_d) \equiv \tau_{dk}$, and $\tau_d$ in turn follows a Dirichlet with prior $\nu$. The generative process we just described defines a joint distribution over the observed data $w$ and unknowns $x = \{\beta, \tau, z\}$ given the hyperparameters $\{\eta, \nu\}$:

$$p(w, x \,|\, \eta, \nu) = \prod_{k=1}^{K} p(\beta_k \,|\, \eta) \times \prod_{d=1}^{D} p(\tau_d \,|\, \nu) \times \prod_{d=1}^{D} \prod_{i=1}^{N} p(w_{di} \,|\, z_{di}, \beta) \, p(z_{di} \,|\, \tau_d), \tag{1}$$

The directed graphical model is given in Fig. 2.

Implementations of approximate inference in LDA include MCMC [15, 26] and variational inference with a mean field approximation [4, 30]. The advantages of our inference approach become clear when it is measured up against the variational mean field algorithm of [4]: first, we make no additional assumptions regarding the model's factorization; second, the number of variational parameters is independent of the size of the corpus, so there is no need to resort to coordinate-wise updates that are typically slow to converge.

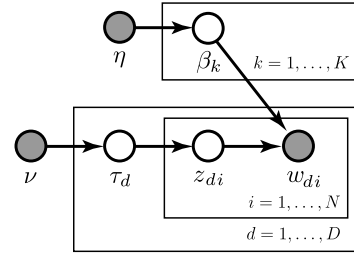

Figure 2: Directed graphical model for LDA. Shaded nodes represent observations or fixed quantities.

## 3 Description of algorithm

The goal is to calculate the expectation of function $\varphi(x)$ with respect to target distribution $p(x)$:

$$E_{p(\cdot)}[\varphi(X)] = \int \varphi(x) \, p(x) \, dx. \tag{2}$$

In LDA, the target density $p(x)$ is the posterior of $x = \{\beta, \tau, z\}$ given $w$ derived via Bayes' rule.

From the importance sampling identity [2], we can obtain an *unbiased* estimate of (2) by drawing $n$ samples from a proposal $q(x)$ and evaluating importance weights $w(x) = p(x)/q(x)$. (Usually $p(x)$ can only be evaluated up to a normalizing constant, in which case the asymptotically unbiased *normalized* importance sampling estimator [2] is used instead.) The Monte Carlo estimator is

$$E_{p(\cdot)}[\varphi(X)] \approx \tfrac{1}{n} \sum_{s=1}^{n} w(x^{(s)}) \, \varphi(x^{(s)}). \tag{3}$$

Unless great care is taken is in designing the proposal $q(x)$, the Monte Carlo estimator will exhibit astronomically high variance for all but the smallest problems.

*Instead, we construct a Monte Carlo estimate* (3) *by replacing $p(x)$ with an alternate target $\hat{p}(x; \theta)$ that resembles it, so that all importance weights are evaluated with respect to this alternate target.* (We elaborate on the exact form of $\hat{p}(x; \theta)$ in Sec. 3.1.) *This new estimator is biased, but we minimize the bias by solving a variational optimization problem.*

Our algorithm has a dual interpretation: it can be interpreted as a stochastic approximation algorithm for solving a variational

- Draw samples from initial density $\hat{p}(x; \theta_1)$.
- **for** $k = 2, 3, 4, \dots$
  - **Stochastic approximation step:** take gradient descent step $\theta_k = \theta_{k-1} - \alpha_k g_k$, where $g_k$ is a Monte Carlo estimate of the gradient of the K-L divergence, and $\alpha_k$ is the variance-safeguarded step size.
  - **SMC step:** update samples and importance weights to reflect new density $\hat{p}(x; \theta_k)$.

Figure 3: Algorithm sketch.

optimization problem, in which the iterates are the parameter vectors $\theta_k$, and it can be equally viewed as a sequential Monte Carlo (SMC) method [12], in which each distribution $\hat{p}(x; \theta_k)$ in the

sequence is chosen dynamically based on samples from the previous iteration. The basic idea is spelled out in Fig. 3. At each iteration, the algorithm selects a new target $\hat{p}(x; \theta_k)$ by optimizing the variational objective. Next, the samples are revised in order to compute the stochastic gradient $g_{k+1}$ at the next iteration. Since SMC is effectively a framework for conducting importance sampling over a sequence of distributions, we describe a "variance safeguard" mechanism (Sec. 3.5) that directly regulates increases in variance at each step by preventing the iterates $\theta_k$ from moving too quickly. It is in this manner that we achieve a trade-off between bias and variance.

Since this is a stochastic approximation method, asymptotic convergence of $\theta_k$ to a minimizer of the objective is guaranteed under basic theory of stochastic approximation [29]. As we elaborate below, this implies that $\hat{p}(x; \theta_k)$ will converge almost surely to the target distribution $p(x)$ as $k$ approaches infinity. And asymptotic variance results from the SMC literature [12] tell us that the Monte Carlo estimates will converge almost surely to the target expectation (2) so long as $\hat{p}(x; \theta_k)$ approaches $p(x)$. A crucial condition is that the stochastic estimates of the gradient be *unbiased*. There is no way to guarantee unbiased estimates under a finite number of samples, so convergence holds only as the number of iterations and number of samples both approach infinity.

To recap, the probabilistic inference recipe we propose has five main ingredients: one, a family of approximating distributions that admits the target (Sec. 3.1); two, a variational optimization problem framed using the K-L divergence measure (Sec. 3.2); three, a stochastic approximation method for finding a solution to the variational optimization problem (Sec. 3.3); four, the implementation of a sequential Monte Carlo method for constructing stochastic estimates of the gradient of the variational objective (Sec 3.4); and five, a way to safeguard the variance of the importance weights at each iteration of the stochastic approximation algorithm (Sec. 3.5).

## 3.1 The family of approximating distributions

The first implementation step is the design of a family of approximating distributions $\hat{p}(x; \theta)$ parameterized by vector $\theta$. In order to devise a useful variational inference procedure, the usual strategy is to restrict the class of approximating distributions to those that factorize in an analytically convenient fashion [4, 19] or, in the dual formulation, to introduce an approximate (but tractable) decomposition of the entropy [32]. Here, we impose no such restrictions on tractability; refer to Fig. 1. We allow any family of approximating distributions so long as it satisfies these three conditions: 1.) there is at least one $\theta = \theta_1$ such that samples can be drawn from $\hat{p}(x; \theta_1)$; 2.) there is a $\theta = \theta^\star$ that recovers the target $\hat{p}(x; \theta^\star) = p(x)$, hence an unbiased estimate of (2); and 3.) the densities are members of the exponential family [13] expressed in standard form

$$\hat{p}(x; \theta) = \exp\{\langle a(x), \theta \rangle - c(\theta)\}, \tag{4}$$

in which $\langle \cdot, \cdot \rangle$ is an inner product, the vector-valued function $a(x)$ is the *statistic* of $x$, and $\theta$ is the *natural* or *canonical* parameterization. The log-normalization factor $c(\theta) \equiv \log \int \exp\langle a(x), \theta \rangle \, dx$ ensures that $\hat{p}(x; \theta)$ represents a proper probability. We further assume that the random vector $x$ can be partitioned into two sets $A$ and $B$ such that it is always possible to draw samples from the conditionals $\hat{p}(x_A \,|\, x_B; \theta)$ and $\hat{p}(x_B \,|\, x_A; \theta)$. Hidden Markov models, mixture models, continuous-time Markov processes, and some Markov random fields are all models that satisfy this condition. This extra condition could be removed without great difficulty, but doing so would add several complications to the description of the algorithm. The restriction to the exponential family is not a strong one as most conventionally-studied densities can be written in the form (4).

For LDA, we chose a family of approximating densities of the form

$$\hat{p}(x; \theta) = \exp \big\{ \textstyle\sum_{d=1}^{D} \sum_{k=1}^{K} (\nu_k + n_{dk} - 1) \log \tau_{dk} + \sum_{k=1}^{K} \sum_{j=1}^{W} (\hat{\eta}_{kj} - 1) \log \beta_{kj}$$
$$+ \phi \textstyle\sum_{k=1}^{K} \sum_{j=1}^{W} m_{kj} \log \beta_{kj} + \gamma \sum_{k=1}^{K} \sum_{j=1}^{W} (c_j - m_{kj}) \log \beta_{kj} - c(\theta) \big\}, \tag{5}$$

where $m_{kj} \equiv \sum_d \sum_i \delta_k(z_{di}) \, \delta_j(w_{di})$ counts the number of times the $j$th word is assigned to the $k$th topic, $n_{dk} \equiv \sum_i \delta_k(z_{di})$ counts the number of words assigned to the $k$th topic in the $d$th document, and $c_j \equiv \sum_d \sum_i \delta_j(w_{di})$ is is the number of times $j$th vocabulary item is observed. The natural parameters are $\theta = \{\hat{\eta}, \phi, \gamma\}$, with $\theta \geq 0$. The target posterior $\hat{p}(x; \theta^\star) \propto p(w, x \,|\, \eta, \nu)$ is recovered by setting $\phi = 1$, $\gamma = 0$ and $\hat{\eta} = \eta$. A sampling density with a tractable expression for $c(\theta)$ is recovered whenever we set $\phi$ equal to $\gamma$. The graphical structure of LDA (Fig. 2) allows us to draw samples from the conditionals $\hat{p}(\beta, \tau \,|\, z; \theta)$ and $\hat{p}(z \,|\, \beta, \tau; \theta)$. Loosely speaking, this choice is meant to strike a balance between the mean field approximation [4] (with parameters $\hat{\eta}_{kj}$) and the tempered distribution (with "local" temperature parameters $\phi$ and $\gamma$).

## 3.2 The variational objective

The Kullback Leibler (K-L) divergence [9] asymmetrically measures the distance between the target distribution $p(x) = \hat{p}(x; \theta^\star)$ and approximating distribution $\hat{p}(x; \theta)$,

$$F(\theta) = \langle E_{\hat{p}(\,\cdot\,;\,\theta)}[a(X)], \theta - \theta^\star \rangle + c(\theta^\star) - c(\theta), \tag{6}$$

the optimal choice being $\theta = \theta^\star$. This is our variational objective. The fact that we cannot compute $c(\theta)$ poses no obstacle to optimizing the objective (6); through application of basic properties of the exponential family, the gradient vector works out to be the matrix-vector product

$$\nabla F(\theta) = \mathrm{Var}_{\hat{p}(\,\cdot\,;\,\theta)}[a(X)](\theta - \theta^\star), \tag{7}$$

where $\mathrm{Var}[a(X)]$ is the covariance matrix of the statistic $a(x)$. The real obstacle is the presence of an integral in (7) that is most likely intractable. With a collection of samples $x^{(s)}$ with importance weights $w^{(s)}$, for $s = 1, \ldots, n$, that approximate $\hat{p}(x; \theta)$, we have the Monte Carlo estimate

$$\nabla F(\theta) \approx \sum_{s=1}^{n} w^{(s)} (a(x^{(s)}) - \bar{a})(a(x^{(s)}) - \bar{a})^T (\theta - \theta^\star), \tag{8}$$

where $\bar{a} \equiv \sum_s w^{(s)} a(x^{(s)})$ denotes the Monte Carlo estimate of the mean statistic. Note that these samples $\{x^{(s)}, w^{(s)}\}$ serve to estimate both the expectation (2) and the gradient (7). The algorithm's performance hinges on a good search direction, so it is worth our while to reduce the variance of the gradient measurements when possible via Rao-Blackwellization [6]. Since we no longer have an exact value for the gradient, we appeal to the theory of stochastic approximation.

## 3.3 Stochastic approximation

Instead of insisting on making progress toward a minimizer of $f(\theta)$ at every iteration, as in gradient descent, stochastic approximation only requires that progress be achieved *on average*. The Robbins-Monro algorithm [28] iteratively adjusts the control variable $\theta$ according to

$$\theta_{k+1} = \theta_k - \alpha_k g_k, \tag{9}$$

where $g_k$ is a noisy observation of $f(\theta_k)$, and $\{\alpha_k\}$ is a sequence of step sizes. Provided the sequence of step sizes satisfies certain conditions, this algorithm is guaranteed to converge to the solution $f(\theta^\star) = 0$; see [29]. In our case, $f(\theta) = \nabla F(\theta) = 0$ is the first-order condition for an unconstrained minimum. Due to poor conditioning, we advocate replacing the gradient descent search direction $\Delta \theta_k = -g_k$ in (9) by the quasi-Newton search direction $\Delta \theta_k = -B_k^{-1} g_k$, where $B_k$ is a damped quasi-Newton (BFGS) approximation of the Hessian [25]. To handle constraints $\theta \geq 0$ introduced in Sec. 3.1, we use the stochastic interior-point method of [5].

After having taken a step along $\Delta \theta_k$, the samples must be updated to reflect the new distribution $\hat{p}(x; \theta_{k+1})$. To accomplish this feat, we use SMC [12] to sample from a sequence of distributions.

## 3.4 Sequential Monte Carlo

In the first step of SMC, samples $x_1^{(s)}$ are drawn from a proposal density $q_1(x) = \hat{p}(x; \theta_1)$ so that the initial importance weights are uniform. After $k$ steps the proposal density is

$$\tilde{q}_k(x_{1:k}) = K_k(x_k \,|\, x_{k-1}) \cdots K_2(x_2 \,|\, x_1)\, \hat{p}(x_1; \theta_1), \tag{10}$$

where $K_k(x' \,|\, x)$ is the Markov kernel that extends the path at every iteration. The insight of [12] is that if we choose the densities $\tilde{p}_k(x_{1:k})$ wisely, we can update the importance weights $\tilde{w}_k(x_{1:k}) = \tilde{p}_k(x_{1:k})/\tilde{q}_k(x_{1:k})$ without having to look at the entire history. This special construction is

$$\tilde{p}_k(x_{1:k}) = L_1(x_1 \,|\, x_2) \cdots L_{k-1}(x_{k-1} \,|\, x_k)\, \hat{p}(x_k; \theta_k), \tag{11}$$

where we've introduced a series of artificial "backward" kernels $L_k(x \,|\, x')$. In this paper, the sequence of distributions is determined by the iterates $\theta_k$, so there remain two degrees of freedom: the choice of forward kernel $K_k(x' \,|\, x)$, and the backward kernel $L_k(x \,|\, x')$. From the assumptions made in Sec. 3.1, a natural choice for the forward transition kernel is the two-stage Gibbs sampler,

$$K_k(x' \,|\, x) = \hat{p}(x'_A \,|\, x'_B; \theta_k)\, \hat{p}(x'_B \,|\, x_A; \theta_k), \tag{12}$$

in which we first draw a sample of $x_B$ (in LDA, the variables $\tau$ and $\beta$) given $x_A$ (the discrete variables $z$), then update $x_A$ conditioned on $x_B$. A Rao-Blackwellized version of the sub-optimal backward kernel [12] then leads to the following expression for updating the importance weights:

$$\tilde{w}_k(x_{1:k}) = \tilde{p}(x_A; \theta_k)/\tilde{p}(x_A; \theta_{k-1}) \times \tilde{w}_{k-1}(x_{1:k-1}), \tag{13}$$

where $x_A$ is the component from time step $k - 1$ restricted to the set $A$, and $\tilde{p}(x_A; \theta_k)$ is the unnormalized version of the marginal $\hat{p}(x_A; \theta_k)$. It also follows from earlier assumptions (Sec 3.1) that it is always possible to compute $\tilde{p}(x_A; \theta)$. Refer to [15] for the marginal of $z$ for LDA.

### 3.5 Safeguarding the variance

A key component of the algorithm is a mechanism that enables the practitioner to regulate the variance of the importance weights and, by extension, the variance of the Monte Carlo estimate of $E[\varphi(X)]$. The trouble with taking a full step (9) is that the Gibbs kernel (12) may be unable to effectively migrate the particles toward the new target, in which case the the importance weights will overcompensate for this failure, quickly leading to a degenerate population. The remedy we propose is to find a step size $\alpha_k$ that satisfies

$$\beta S_k(\theta_k) \leq S_{k-1}(\theta_{k-1}), \tag{14}$$

for $\beta \in [0, 1]$, whereby a $\beta$ near 1 leads to a stringent safeguard, and we've defined

$$S_k(\theta_k) \equiv \sum_{s=1}^{n} (\tilde{w}_k(x_{1:k}^{(s)}) - \tfrac{1}{n})^2 \tag{15}$$

---

- Let $n$, $\theta_1$, $\theta^\star$, $A$, $B$, $\{\alpha_k\}$ be given.
- Draw $x^{(s)} \sim \hat{p}(x; \theta_1)$, set $w^{(s)} = 1/n$.
- Set inverse Hessian $H$ to the identity.
- **for** $k = 2, 3, 4, \dots$
  1. Compute $g_k \approx \nabla F(\theta_{k-1})$; see (8).
  2. **if** $k > 2$, **then** modify $H$ following damped quasi-Newton update.
  3. Compute variance-safeguarded step size $\alpha_k \leq \hat{\alpha}_k$ given $\Delta\theta_k = -Hg_k$.
  4. Set $\theta_k = \theta_{k-1} + \alpha_k \Delta\theta_k$.
  5. Update $w^{(s)}$ following (13).
  6. Run the two-stage Gibbs sampler:
     - Draw $x_B^{(s)} \sim \hat{p}(\cdot \,|\, x_A^{(s)}; \theta_k)$.
     - Draw $x_A^{(s)} \sim \hat{p}(\cdot \,|\, x_B^{(s)}; \theta_k)$.
  7. Resample particles, if necessary.

Figure 4: The proposed algorithm.

---

to be the sample variance ($\times n$) for our choice of $L(x \,|\, x')$. Note that since our variance safeguard scheme is myopic, the behaviour of the algorithm can be sensitive to the number of iterations.

The safeguarded step size is derived as follows. The goal is to find the largest step size $\alpha_k$ satisfying (14). Forming a Taylor-series expansion with second-order terms about the point $\alpha_k = 0$, the safeguarded step size is the solution to

$$\tfrac{1}{2}\Delta\theta_k^T \nabla^2 S_k(\theta_{k-1})\Delta\theta_k \alpha_k^2 + \Delta\theta_k^T \nabla S_k(\theta_{k-1})\,\alpha_k = \tfrac{1-\beta}{\beta} S_{k-1}(\theta_{k-1}), \tag{16}$$

where $\Delta\theta_k$ is the search direction at iteration $k$. In our experience, the quadratic approximation to the importance weights (13) was unstable as it occasionally recommended strange step sizes, but a naive importance weight update without Rao-Blackwellization yielded a reliable bound on (14). The derivatives of $S_k(\theta_k)$ work out to sample estimates of second and third moments that can be computed in $O(n)$ time. Since the importance weights initially have zero variance, no positive step size will satisfy (14). We propose to also permit step sizes that do not drive the ESS below a factor $\xi \in (0, 1)$ from the optimal sample. Resampling will still be necessary over long sequences to prevent the population from degenerating. The basic algorithm is summarized in Fig. 4.

## 4 Application to population genetics

Microsatellite genetic markers have been used to determine the genealogy of human populations, and to assess individuals' ancestry in inferring disease risks [16]. The problem is that all these tasks require defining *a priori* population structure. The Bayesian model of Pritchard et al. [26] offers a solution to this conundrum by simultaneously identifying both patterns of population subdivision and the ancestry of individuals from highly

| text corpus | | population structure |
|---|---|---|
| documents | | individuals |
| topics | $\Leftrightarrow$ | demes |
| languages | | loci |
| vocabulary | | alleles |

Figure 5: Correspondence between LDA [4] and the population structure [26] models.

variable genetic markers. This model is the same as LDA assuming fixed Dirichlet priors and a single genetic marker; see Fig. 5 for the connection between the two domains. This model, however, can be frustrating to work with because independent MCMC simulations can produce remarkably different answers for the same data, even simulations millions of samples long. Such inference challenges have been observed in other mixture models [7]; MCMC can do a poor job exploring the hypothesis space when there are several divergent hypotheses that explain the data.

**Method.** We used the software CoaSim [21] to simulate the evolution of genetic markers following a coalescent process. The coalescent is a lineage of alleles in a sample traced backward in time to their common ancestor allele, and the coalescent process is the stochastic process that generates the genealogy [17]. We introduced divergence events at various coalescent times (see Fig. 6) so that we ended up with 4 isolated populations. We simulated 10 microsatellite markers each with a maximum of 30 alleles. We simulated the markers twice with scaled mutation rates of 2 and $\frac{1}{2}$, and for each rate we simulated 60 samples from the coalescent process (15 diploid individuals from each of the 4 populations). These samples are the words $w$ in LDA. This may not seem like a large data set, but it will be large enough to impose major challenges to approximate inference.

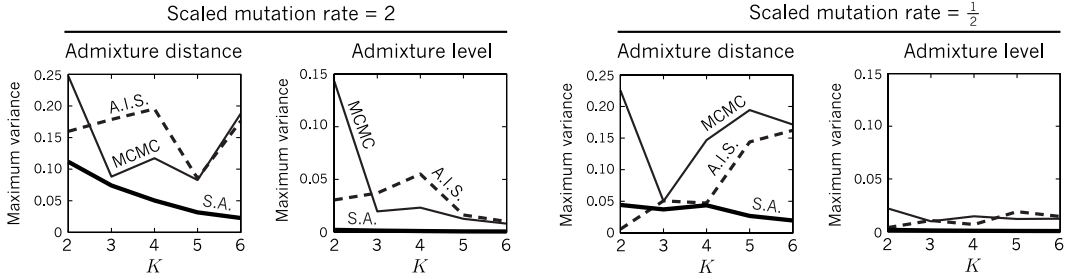

Figure 7: Variance in estimates of the admixture distance and admixture level taken over 20 trials.

The goal is to obtain posterior estimates that re-
cover the correct population structure (Fig. 6) and
exhibit high agreement in independent simula-
tions. Specifically, the goal is to recover the mo-
ments of two statistics: the *admixture distance*, a
measure of two individuals' dissimilarity in their
ancestry, and the *admixture level* where 0 means
an individual's alleles all come from a single pop-
ulation, and 1 means its ancestry is shared equally
among the $K$ populations. The admixture dis-
tance between individuals $d$ and $d'$ is

$$\varphi(\tau_d, \tau_{d'}) \equiv \tfrac{1}{2} \sum_{k=1}^{K} |\tau_{dk} - \tau_{d'k}|, \qquad (17)$$

and the admixture level of the $d$th individual is

$$\psi(\tau_d) \equiv 1 - \tfrac{K}{2(K-1)} \sum_{k=1}^{K} \left| \tau_{dk} - \tfrac{1}{K} \right|. \quad (18)$$

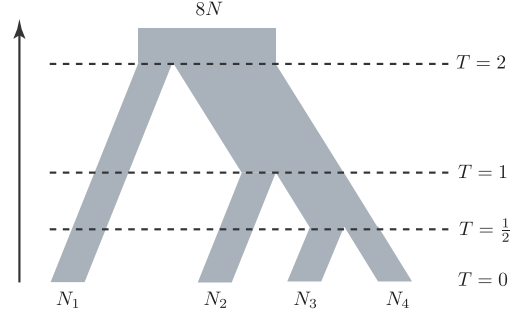

Figure 6: The structured coalescent process
with divergence events at coalescent times $T =
0, \frac{1}{2}, 1, 2$. The width of the branches represents
effective population size, and the arrow points
backward in time. The present isolated popu-
lations are labeled left-to-right 1 through 4.

We compared our algorithm to MCMC as implemented in the software STRUCTURE [26], and to
another SMC algorithm, annealed importance sampling (AIS) [24], with a uniform tempering
schedule. One possible limitation of our study is that the choice of temperature scehdule can be
critical to the success of AIS, and we did not thoroughly investigate alternative schedules. Also,
note that our intent was not to present an exhaustive comparison of Monte Carlo methods, so we
did not compare to population MCMC [18], for example, which has advantages similar to AIS.

For the two data sets, and for each $K$ from 2 to 6 (the most appropriate setting being $K = 4$), we
carried out 20 independent trials of the three methods. For fair comparison, we ran the methods
with the same number of sampling events: for MCMC, a Markov chain of length 50,000 and
burn-in of 10,000; for both SMC methods, 100 particles and 500 iterations. Additional settings
included an ESS threshold of 50, maximum step sizes $\alpha_k = 1/(1 + k)^{0.6}$, centering parameters
$\sigma_k = 1/k^{0.9}$ for the stochastic interior-point method, safeguards $\beta = 0.95$ and $\xi = 0.9$, and a
quasi-Newton damping factor of 0.75. We set the initial iterate of stochastic approximation to
$\phi = \gamma = \hat{\eta}_{kj} = \eta_j^\star$. We used uniform Dirichlet priors $\eta_j^\star = \nu_k = 0.1$ throughout.

**Results.** First let's examine the variance in the answers. Fig. 7 shows the variance in the estimates
of the admixture level and admixture distance over the independent trials. To produce these plots,
at every $K$ we took the individual $d$ or pair $(d, d')$ that exhibited the most variance in the estimate
of $E[\varphi(\tau_d, \tau_{d'})]$ and $E[\psi(\tau_d)]$. What we observe is that the stochastic approximation method
produced significantly more consistent estimates in almost all cases, whereas AIS offered little or
no improvement over MCMC. The next step is to examine the accuracy of these answers.

Fig. 8 shows estimates from MCMC and stochastic approximation selected trials under a mutation
rate of $\frac{1}{2}$ and $K = 4$ *(left-hand side)*, and under a mutation rate of 2 and $K = 3$ *(right-hand side)*.
The trials were chosen to reflect the extent of variation in the answers. The mean and standard
deviation of the admixture distance statistic are drawn as matrices. The 60 rows and 60 columns in
each matrix correspond to individuals sorted by their true population label; the rows and columns
are ordered so that they correspond to the populations 1 through 4 in Fig. 6. In each "mean" matrix,
a light square means that two individuals share little ancestry in common, and a dark square means
that two individuals have similar ancestry. In each "std. dev." matrix, the darker the square, the
higher the variance. In the first trial *(top-left)*, the MCMC algorithm mostly recovered the correct

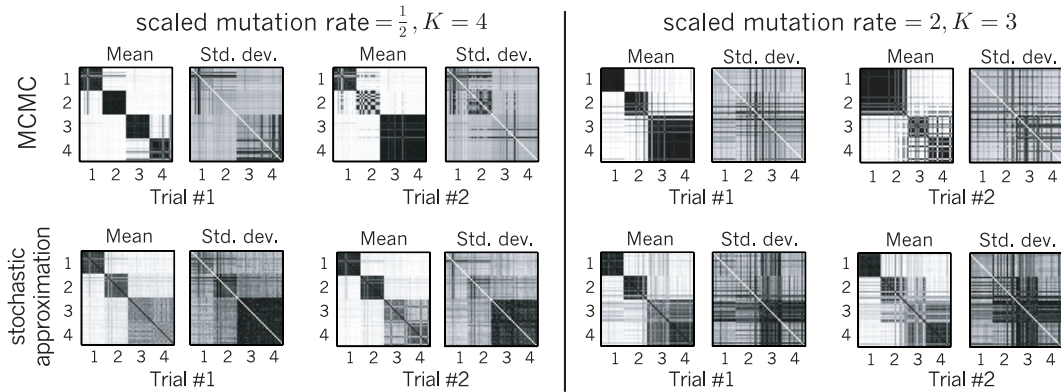

Figure 8: Estimated mean and standard deviation ("std. dev.") of the admixture distance statistic for two independent trials and at two different simulation settings. See the text for a full explanation.

population structure; *i.e.* it successfully assigned individuals to their coalescent populations based on the sampled alleles $w$. As expected, the individuals from populations 3 and 4 were hardest to distinguish, hence the high standard deviation in the bottom-right entries of the matrix. The results of the second trial are less satisfying: MCMC failed to distinguish between individuals from populations 3 and 4, and it decided rather arbitrarily to partition the samples originating from population 2. In all these experiments, AIS exhibited behaviour that was very similar to MCMC.

Under the same conditions, our algorithm *(bottom-left)* failed to distinguish between the third and fourth populations. The trials, however, are more consistent and do not mislead by placing high confidence in these answers; observe the large number of dark squares in the bottom-right portion of the "std. dev." matrix. This evidence suggests that these trials are more representative of the true posterior because the MCMC trials are inconsistent and occasionally spurious (trial #2). This trend is repeated in the more challenging inference scenario with $K = 3$ and a mutation rate of 2 *(right-hand side)*. MCMC, as before, exhibited a great deal of variance in its estimates of the admixture distance: the estimates from the first trial are very accurate, but the second trial strangely failed to distinguish between populations 1 and 2, and did not correctly assign the individuals in populations 3 and 4. What's worse, MCMC placed disproportionate confidence in these estimates. The stochastic approximation method also exhibited some variance under these conditions, but importantly it did not place nearly so much confidence in its solutions; observe the high standard deviation in the matrix entries corresponding to the individuals from population 3.

## 5    Conclusions and discussion

In this paper, we proposed a new approach to probabilistic inference grounded on variational, Monte Carlo and stochastic approximation methodology. We demonstrated that our sophisticated method pays off in terms of producing more consistent, reliable estimates for a real and challenging inference problem in population genetics. Some of the components such as the variance safeguard have not been independently validated, so we cannot fully attest to how critical they are, at least beyond the motivation we already gave. More standard tricks, such as Rao-Blackwellization, were explicitly included to demonstrate that well-known techniques from the Monte Carlo literature apply without modification to our algorithm. We have argued for the generality of our inference approach, but ultimately the success of our scheme hinges on a good choice of the variational approximation. Thus, it remains to be seen how well our results extend to probabilistic graphical models beyond LDA, and how much ingenuity will be required to achieve favourable outcomes.

Another critical issue, as we mentioned in Sec. 3.5, is the sensitivity of our method to the number of iterations. This issue is related to the bias-variance trade-off, and in the future we would like to explore more principled ways to formulate this trade-off, in the process reducing this sensitivity.

### Acknowledgments

We would like to thank Matthew Hoffman, Nolan Kane, Emtiyaz Khan, Hendrik Kück and Pooja Viswanathan for their input, and the reviewers for exceptionally detailed and thoughtful comments.

# References

[1] C. Andrieu and E. Moulines. On the ergodicity properties of some adaptive MCMC algorithms. *Annals of Applied Probability*, 16:1462–1505, 2006.

[2] C. Andrieu, N. de Freitas, A. Doucet, and M. I. Jordan. An introduction to MCMC for machine learning. *Machine Learning*, 50:5–43, 2003.

[3] M. J. Beal. *Variational Algorithms for Approximate Bayesian Inference*. PhD thesis, University College London, 2003.

[4] D. Blei, A. Y. Ng, and M. I. Jordan. Latent Dirichlet allocation. *Journal of Machine Learning Research*, 3:993–1022, 2003.

[5] P. Carbonetto, M. Schmidt, and N. de Freitas. An interior-point stochastic approximation method and an L1-regularized delta rule. In *Advances in Neural Information Processing Systems*, volume 21. 2009.

[6] G. Casella and C. P. Robert. Rao-Blackwellisation of sampling schemes. *Biometrika*, 83:81–94, 1996.

[7] G. Celeux, M. Hurn, and C. P. Robert. Computational and inferential difficulties with mixture posterior distributions. *Journal of the American Statistical Association*, 95:957–970, 2000.

[8] D. W. Coltman. Molecular ecological approaches to studying the evolutionary impact of selective harvesting in wildlife. *Molecular Ecology*, 17:221–235, 2007.

[9] T. M. Cover and J. A. Thomas. *Elements of Information Theory*. Wiley, 1991.

[10] P.-T. de Boer, D. P. Kroese, S. Mannor, and R. Y. Rubinstein. A tutorial on the cross-entropy method. *Annals of Operations Research*, 134:19–67, 2005.

[11] N. de Freitas, P. Højen-Sørensen, M. I. Jordan, and S. Russell. Variational MCMC. In *Proceedings of the 17th Conference on Uncertainty in Artificial Intelligence*, pages 120–127, 2001.

[12] P. Del Moral, A. Doucet, and A. Jasra. Sequential Monte Carlo samplers. *Journal of the Royal Statistical Society*, 68:411–436, 2006.

[13] A. J. Dobson. *An Introduction to Generalized Linear Models*. Chapman & Hall/CRC Press, 2002.

[14] A. Doucet, S. Godsill, and C. Andrieu. On sequential Monte Carlo sampling methods for Bayesian filtering. *Statistics and Computing*, 10:197–208, 2000.

[15] T. L. Griffiths and M. Steyvers. Finding scientific topics. *Proceedings of the National Academy of Sciences*, 101:5228–5235, 2004.

[16] D. L. Hartl and A. G. Clark. *Principles of population genetics*. Sinauer Associates, 2007.

[17] J. Hein, M. H. Schierup, and C. Wiuf. *Gene genealogies, variation and evolution: a primer in coalescent theory*. Oxford University Press, 2005.

[18] A. Jasra, D. Stephens, and C. Holmes. On population-based simulation for static inference. *Statistics and Computing*, 17:263–279, 2007.

[19] M. Jordan, Z. Ghahramani, T. Jaakkola, and L. Saul. An introduction to variational methods for graphical models. In M. Jordan, editor, *Learning in Graphical Models*, pages 105–161. MIT Press, 1998.

[20] F. Liang, C. Liu, and R. J. Carroll. Stochastic approximation in Monte Carlo computation. *Journal of the American Statistical Association*, 102:305–320, 2007.

[21] T. Mailund, M. Schierup, C. Pedersen, P. Mechlenborg, J. Madsen, and L. Schauser. CoaSim: a flexible environment for simulating genetic data under coalescent models. *BMC Bioinformatics*, 6, 2005.

[22] T. Minka. *A family of algorithms for approximate Bayesian inference*. PhD thesis, MIT, 2001.

[23] R. Neal and G. Hinton. A view of the EM algorithm that that justifies incremental, sparse, and other variants. In M. Jordan, editor, *Learning in graphical models*, pages 355–368. Kluwer Academic, 1998.

[24] R. M. Neal. Annealed importance sampling. *Statistics and Computing*, 11:125–139, 2001.

[25] M. J. D. Powell. Algorithms for nonlinear constraints that use Lagrangian functions. *Mathematical Programming*, 14:224–248, 1978.

[26] J. K. Pritchard, M. Stephens, and P. Donnelly. Inference of population structure using multilocus genotype data. *Genetics*, 155:945–959, 2000.

[27] P. Ravikumar. *Approximate Inference, Structure Learning and Feature Estimation in Markov Random Fields*. PhD thesis, Carnegie Mellon University, 2007.

[28] H. Robbins and S. Monro. A stochastic approximation method. *Annals of Math. Statistics*, 22, 1951.

[29] J. C. Spall. *Introduction to stochastic search and optimization*. Wiley-Interscience, 2003.

[30] Y. W. Teh, D. Newman, and M. Welling. A collapsed variational Bayesian inference algorithm for latent Dirichlet allocation. In *Advances in Neural Information Processing Systems*, volume 19, 2007.

[31] M. J. Wainwright. *Stochastic processes on graphs with cycles: geometric and variational approaches*. PhD thesis, Massachusetts Institute of Technology, 2002.

[32] J. S. Yedidia, W. T. Freeman, and Y. Weiss. Constructing free-energy approximations and generalized belief propagation algorithms. *IEEE Transactions on Information Theory*, 51:2282–2312, 2005.

[33] L. Younes. Stochastic gradient estimation strategies for Markov random fields. In *Proceedings of the Spatial Statistics and Imaging Conference*, 1991.

